# Ordered Classes and Incomplete Examples in Classification

**Mark Mathieson**
Department of Statistics, University of Oxford
1 South Parks Road, Oxford OX1 3TG, UK
E-mail: mathies@stats.ox.ac.uk

## Abstract

The classes in classification tasks often have a natural ordering, and the training and testing examples are often incomplete. We propose a non-linear ordinal model for classification into ordered classes. Predictive, simulation-based approaches are used to learn from past and classify future incomplete examples. These techniques are illustrated by making prognoses for patients who have suffered severe head injuries.

## 1 Motivation

Jennett *et al.* (1979) reported data on patients with severe head injuries. For each patient some of the information in Table 1 was available shortly after injury. The objective is to predict the degree of recovery attained within six months as measured by outcome. This problem exhibits two characteristics that are common in classification tasks: allocation of examples into classes which have a natural ordering, and learning from past and classifying future incomplete examples.

## 2 A Flexible Model for Ordered Classes

The Bayes decision rule (see, for example, Ripley, 1996) depends on the loss $L(j, k)$ incurred in assigning to class $k$ an object belonging to class $j$. When better information is unavailable, for unordered or *nominal* classes we treat every mis-classification as equally serious: $L(j, k)$ is 0 when $j = k$ and 1 otherwise. For ordered classes, when the $K$ classes are numbered from 1 to $K$ in their natural order, a better default choice is $L(j, k) = \mid j - k \mid$. A class is then given support by its position in the ordering, and the Bayes rule will sometimes assign patterns to classes that do not have maximum posterior probability to avoid making a serious error.

**Table 1**: Definition of variables with proportion missing.

| Variable | Definition | Missing % |
|---|---|---|
| age | Age in decades (1=0–9, 2=10–19, . . . , 8=70+). | 0 |
| emv | Measure of eye, motor and verbal response to stimulation (1–7). | 41 |
| motor | Motor response patterns for all limbs (1–7). | 33 |
| change | Change in neurological function over the first 24 hours (–1,0,+1). | 78 |
| eye | Eye indicant. 1 (bad), 2 (impaired), 3 (good). | 65 |
| pupils | Pupil reaction to light. 1 (non-reacting), 2 (reacting). | 30 |
| outcome | Recovery after six months based on Glasgow Outcome Scale. 1 (dead/vegetative), 2 (severe disability), 3 (moderate/good recovery). | 0 |

If the classes in a classification problem are ordered the ordering should also be reflected in the probability model. Methods for nominal tasks can certainly be used for ordinal problems, but an ordinal model should have a simpler parameterization than comparable nominal models, and interpretation will be easier. Suppose that an example represented by a row vector $X$ belongs to class $C = C(X)$. To make the Bayes-optimal classification it is sufficient to know the posterior probabilities $p(C = k \mid X = x)$. The *ordinal logistic regression* (OLR) model for $K$ ordered classes models the cumulative posterior class probabilities $p(C \leqslant k \mid X = x)$ by

$$\log \left[ \frac{p(C \leqslant k \mid X = x)}{1 - p(C \leqslant k \mid X = x)} \right] = \phi_k - \eta(x) \qquad k = 1, \ldots, K - 1, \tag{1}$$

for some function $\eta$. We impose the constraints $\phi_1 \leqslant \ldots \leqslant \phi_{K-1}$ on the *cut-points* to ensure that $p(C \leqslant k \mid X = x)$ increases with $k$. If $\phi_0 = -\infty$ and $\phi_K = \infty$ then (1) gives

$$p(C = k \mid X = x) = \sigma(\phi_k - \eta(x)) - \sigma(\phi_{k-1} - \eta(x)) \qquad k = 1, \ldots, K$$

where $\sigma(x) = 1/(1 + e^{-x})$. McCullagh (1980) proposed linear OLR where $\eta(x) = x\beta$.

The posterior probabilities depend on the patterns $x$ only through $\eta$, and high values of $\eta(x)$ correspond to higher predicted classes (Figure 1a). This can be useful for interpreting the fitted model. However, linear OLR is rather inflexible since the decision boundaries are always parallel hyperplanes. Departures from linearity can be accommodated by allowing $\eta$ to be a non-linear function of the feature space. We extend OLR to non-linear ordinal logistic regression (NOLR) by letting $\eta(x)$ be the single linear output of a feed-forward neural network with input vector $x$, having skip-layer connections and sigmoid transfer functions in the hidden layer (Figure 1b). Then for weights $w_{ij}$ and biases $b_j$ we have

$$\eta(x) = \sum_{i \to o} w_{io} x_{(i)} + \sum_{j \to o} w_{jo} \sigma(b_j + \sum_{i \to j} w_{ij} x_{(i)}),$$

where $\sum_{i \to j}$ denotes the sum over $i$ such that node $i$ is connected to node $j$, and node $o$ is the single output node. The usual output-unit bias is incorporated in the cut-points. Observe that OLR is the special case of NOLR with no hidden nodes. Although the network component of NOLR is a universal approximator the NOLR model cannot approximate all probability densities arbitrarily well (unlike 'softmax', the most similar nominal method).

The likelihood for the cut-points $\phi = (\phi_1, \ldots, \phi_{K-1})$ and network parameters $\mathbf{w}$ given a training set $\mathcal{T} = \{(x_i, c_i) \mid i = 1, \ldots, n\}$ of $n$ correctly classified examples is

$$\ell(\mathbf{w}, \phi) = \prod_{i=1}^{n} p(c_i \mid x_i) = \prod_{i=1}^{n} \left[ \sigma(\phi_{c_i} - \eta(x_i; \mathbf{w})) - \sigma(\phi_{c_i-1} - \eta(x_i; \mathbf{w})) \right]. \tag{2}$$

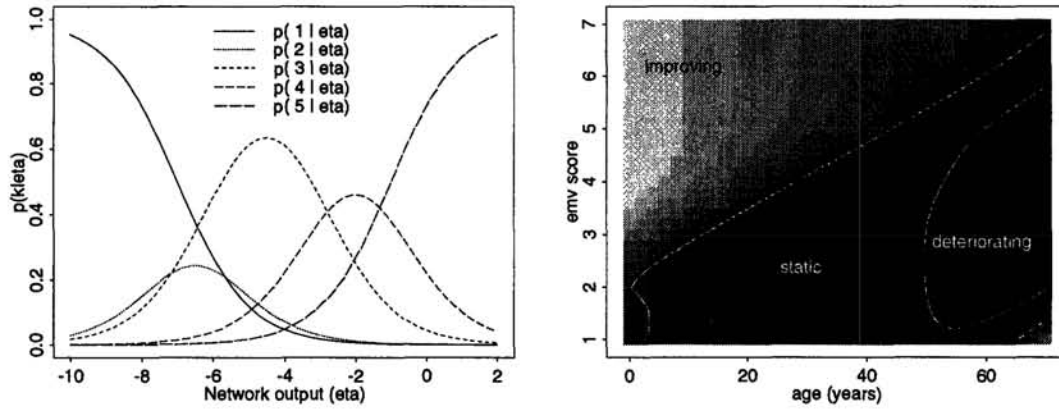

**Figure 1**: **(a)** $p(k \mid \eta)$ plotted against $\eta$ for an OLR model with $K = 5$ classes and $\phi = (-7, -6, -3, -1)$. **(b)** The network output $\eta(x)$ from a NOLR model used to predict change given all other variables (except outcome) predicts that young patients with high emv score are likely to improve over first 24 hours. While age and emv are varied, other variables are fixed. Dark shading denotes low values of $\eta(x)$. The Bayes decision boundaries are shown for loss $L(j, k) = \mid j - k \mid$.

If we estimate the classifier by substituting the maximum likelihood estimates we must maximize (2) whilst constraining the cut-points to be increasing (Mathieson, 1996). To avoid over-fitting we regularize both by weight decay (which is equivalent to putting independent Gaussian priors on the network weights) and by imposing independent Gamma priors on the differences between adjacent cut-points. The minimand is now $-\log \ell(\mathbf{w}, \phi) + \lambda D(\mathbf{w}) + E(\phi; t, \alpha)$ with hyperparameters $\lambda > 0, t, \alpha$ (to be chosen by cross-validation, for example, or averaged over under a Bayesian scheme) where $D(\mathbf{w}) = \sum_{i,j} w_{ij}^2$ and

$$E(\phi) = \sum_{i=2}^{K-1} [t(\phi_i - \phi_{i-1}) + (1 - \alpha) \log(\phi_i - \phi_{i-1})] .$$

## 3   Classification and Incomplete Examples

We now consider simulation-based methods for training diagnostic paradigm classifiers from incomplete examples, and classifying future incomplete examples. To avoid modelling the missing data we assume that the missing data mechanism is independent of the missing values given the observed values (*missing at random*) and that the missing data and data generation mechanisms are independent (*ignorable*) (Little & Rubin, 1987). This assumption is rarely true but is usually less damaging than adopting crude *ad hoc* approaches to missing values.

### 3.1   Learning from Incomplete Examples

The training set is $\mathcal{T} = \{(x_i^o, c_i) \mid i = 1, \ldots, n\}$ where $x_i^o, x_i^u$ are the observed and unobserved parts of the $i$th example, which belongs to class $c_i$. Define $\mathcal{X}^o = \{x_i^o \mid i = 1, \ldots, n\}$ and $\mathcal{X}^u = \{x_i^u \mid i = 1, \ldots, n\}$, and use $\mathcal{C}$ to denote all the classes, so $\mathcal{T} = (\mathcal{X}^o, \mathcal{C})$. We assume that $\mathcal{C}$ is fully observed. Under the diagnostic paradigm (which includes logistic regression and its non-linear and ordinal variants such as 'softmax' and

NOLR) we model $p(c \mid x)$ by $p(c \mid x; \theta)$ giving the conditional likelihood

$$\ell(\theta) = \prod_{i=1}^{n} p(c_i \mid x_i^o; \theta) = \prod_{i=1}^{n} \mathbb{E}_{X_i^u \mid x_i^o} p(c_i \mid x_i^o, X_i^u; \theta) = \mathbb{E}_{\mathcal{X}^u \mid \mathcal{X}^o} \prod_{i=1}^{n} p(c_i \mid x_i^o, X_i^u; \theta) \tag{3}$$

when the examples are independent. The model for $p(c \mid x)$ contains no information about $p(x)$ and so we construct a model for $p(x^u \mid x^o)$ separately using $\mathcal{T}$ (Section 3.2). Once we can sample $x_{i1}^u, \ldots, x_{im}^u$ from $p(x_i^u \mid x_i^o, c_i)$ a Monte Carlo approximation for $\ell(\theta)$ based on the last expression of (3) by averaging over repeated imputations of the missing values in the training set (Little & Rubin, 1987, and earlier):

$$\log \ell(\theta) \approx \log \left( \frac{1}{m} \sum_{j=1}^{m} \prod_{i=1}^{n} p(c_i \mid x_i^o, x_{ij}^u; \theta) \right). \tag{4}$$

Existing algorithms for finding maximum likelihood estimates for $\theta$ allow maximization of the individual summands in (4) with respect to $\theta$, but in general the software will require extensive modification in order to maximize the average. This problem can be avoided if we approximate the arithmetic average over the imputations by a geometric one so that $\ell(\theta) \approx \left( \prod_j \prod_i p(c_i \mid x_i^o, x_{ij}^u; \theta) \right)^{1/m}$. Now the log-posterior averages over the *log* of the likelihoods of the completed training sets, so standard estimation algorithms can be used on a training set formed by pooling all completions of the training set, giving each weight $1/m$. The approximation $\log \frac{1}{m} \sum_j p_j \approx \frac{1}{m} \sum_j \log p_j$ has been made, where we define $p_j(\theta) = \prod_i p(c_i \mid x_i^o, x_{ij}^u; \theta)$, although in fact $\log \frac{1}{m} \sum_j p_j \geqslant \frac{1}{m} \sum_j \log p_j$ everywhere. Suppose that the $p_j$ are well approximated by some function $p$ for the region of interest in the parameter space. Then in this region

$$\log \frac{1}{m} \sum_j p_j - \frac{1}{m} \sum_j \log p_j \approx \frac{1}{2m} \sum_i \left( \frac{p_i - p}{p} \right)^2 - \frac{1}{2m^2} \sum_{i,j} \frac{(p_i - p)(p_j - p)}{p^2} \tag{5}$$

and so the approximation will be reasonable when the imputed values have little effect on the likelihood of the completed training sets. Note that the approximation cannot be improved by increasing $m$; (5) does not tend to zero as $m \to \infty$. The relative effects on the likelihood of making this approximation and the Monte Carlo approximation (4) will be problem specific and dependent on $m$.

The predictive approach (Ripley, 1996, for example) incorporates uncertainty in $\theta$ by estimating $p(c \mid x)$ as $\tilde{p}(c \mid x) = \mathbb{E}_{\theta \mid \mathcal{T}} p(c \mid x; \theta)$. Changing the order of integration gives

$$\tilde{p}(c \mid x) = \int p(c \mid x; \theta) p(\theta \mid \mathcal{T}) \, d\theta \propto \int p(c \mid x; \theta) p(\theta) \prod_{i=1}^{n} \mathbb{E}_{X^u \mid x_i^o} p(c_i \mid x_i^o, X_i^u; \theta) \, d\theta$$

$$= \mathbb{E}_{\mathcal{X}^u \mid \mathcal{X}^o} \int p(c \mid x; \theta) p(\theta) \prod_{i=1}^{n} p(c_i \mid x_i^o, X_i^u; \theta) \, d\theta \tag{6}$$

This justifies applying standard techniques for complete data to build a separate classifier using each completed training set, and then averaging the posterior class probabilities that they predict. The integral over $\theta$ in (6) will usually require approximation; in particular we could average over plug-in estimates to obtain $\tilde{p}(c \mid x) \approx \frac{1}{m} \sum_{j=1}^{m} p(c \mid x; \hat{\theta}_j)$, where $\hat{\theta}_j$ is the MAP estimate of $\theta$ based only on the $j$th imputed training set. A more subtle approach

**Table 2**: Classifier performance on 301 complete test examples. See Section 4.

| Training set | Test set loss |
|---|---|
| 40 complete training examples only | 132 |
| 40 complete + 206 incomplete training examples: | |
| • Median imputation (In each variable, substitute the median for missing values whenever they occur.) | 149 |
| • Averaging predicted probabilities over 1000 completions of $\mathcal{T}$ generated by: | |
| ▷ Unconditional imputation (Sample missing values from the empirical distribution of each variable in the training set.) | 133 |
| ▷ Gibbs sampling from $p(\mathcal{X}^u \mid \mathcal{X}^o, \hat{\psi})$ | 118 |
| Pool the 1000 completions from the line above to form a single training set | 117 |

(Ripley, 1994) approximates each posterior by a mixture of Gaussians centred at the local maxima $\hat{\theta}_{j1}, \ldots, \hat{\theta}_{jR_j}$ of $p(\theta \mid \mathcal{T}, \mathcal{X}_j^u)$ to give

$$p(\theta \mid \mathcal{T}, \mathcal{X}_j^u) \approx \frac{1}{\sum_r w_{jr}} \sum_{r=1}^{R_j} w_{jr} \mathrm{N}(\theta; \hat{\theta}_{jr}, H_{jr}^{-1}) \tag{7}$$

where: $\mathrm{N}(\cdot; \mu, \Sigma)$ is the Gaussian density function with mean $\mu$ and covariance matrix $\Sigma$, the Hessian $H_{jr} = \frac{\partial^2}{\partial \theta^T \partial \theta} \log p(\theta \mid \mathcal{T}, \mathcal{X}_j^u)$ is evaluated at $\hat{\theta}_{jr}$ and, using Laplace's approximation, $w_{jr} = p(\hat{\theta}_{jr} \mid \mathcal{T}, \mathcal{X}_j^u) \mid H_{jr} \mid^{-1/2}$. We can average over the maxima to get $\tilde{p}(c \mid x) \approx (m \sum_{j,r} w_{jr})^{-1} \sum_{j,r} p(c \mid x; \hat{\theta}_{jr})$, but the full-blooded approach samples from the 'mixture of mixtures' approximation to $p(\theta \mid \mathcal{T})$ and also uses importance sampling to compute the predictive estimates $\tilde{p}$.

## 3.2 The Imputation Model

We need samples from $p(x_i^u \mid x_i^o, c_i)$ for each $i$. When many patterns of missing values occur it is not practical to model $p(x^u \mid x^o, c)$ for each pattern, but Markov chain Monte Carlo methods can be employed. The Gibbs sampler is convenient and in its most basic form requires models for the distribution of each element of $x$ given the others, that is $p(x^{(j)} \mid x^{(-j)}, c)$ where $x^{(-j)} = (x^{(1)}, \ldots, x^{(j-1)}, x^{(j+1)}, \ldots, x^{(p)})$. We model these *full conditionals* parametrically as $p(x^{(j)} \mid x^{(-j)}, c; \psi)$ and assume here that the parameters for each of the full conditionals are disjoint, so $p(x^{(j)} \mid x^{(-j)}, c; \psi^{(j)})$ where $\psi = (\psi^{(1)}, \ldots, \psi^{(p)})$. When $x^{(j)}$ takes discrete values this is a classification task, and for continuous values a regression problem. Under certain conditions the chain of dependent samples of $X^u$ converges in distribution to $p(x^u \mid x^o, \psi)$ and the ergodic average of $p(c \mid x^o, X^u)$ converges as required to the predictive estimate $\tilde{p}(c \mid x^o)$. We usually take every $w$th sample to provide a cover of the space in fewer samples, reducing the computation required to learn the classifier. It is essential to check convergence of the Gibbs sampler although we do not give details here.

If we have sufficient complete examples we might use them to estimate $\psi$ to be $\hat{\psi}$ and Gibbs sample from $p(\mathcal{X}^u \mid \mathcal{X}^o; \hat{\psi})$. Otherwise, in the Bayesian framework, incorporate $\psi$ into the sampling scheme by Gibbs sampling from $p(\psi, \mathcal{X}^u \mid \mathcal{X}^o)$ (the solution suggested by Li, 1988). In the head injury example we report results using the former approach. (The latter was found to make little improvement and requires considerably more computation time.)

**Table 3**: Predictive approximations for a NOLR model fitted to a single completion $\mathcal{T}, \mathcal{X}^u$ of the training set. The likelihood maxima at $\hat{\theta}_1$ and $\hat{\theta}_2$ account for over 0.99 of the posterior probability.

|  | $\hat{\theta}_1$ | $\hat{\theta}_2$ | |
|---|---|---|---|
| Posterior probability | 0.929 | 0.071 | |
| $-\log p(\hat{\theta}_i \mid \mathcal{T}, \mathcal{X}^u)$ | 176.10 | 174.65 | |
| Test set loss: | | | **Predictive:** |
| • using the plug-in classifier $p(c \mid x; \hat{\theta}_i)$ | 128 | 149 | 126 |
| • averaging over 10,000 samples from Gaussian | 120 | 137 | 119 |

### 3.3 Classifying Incomplete Examples

We could build a separate classifier for each pattern of missing data that occurs, but this can be computationally expensive, will lose information and the classifiers need not make consistent predictions. We know that $p(c \mid x^o) = \mathbb{E}_{X^u \mid x^o} p(c \mid x^o, X^u)$ so it seems better to classify $x^o$ by averaging over repeated imputations of $x^u$ from the imputation model.

## 4 Prognosis After Head Injury

We now return to the head injury prognosis example to learn a NOLR classifier from a training set containing 40 complete and 206 incomplete examples. The NOLR architecture (4 nodes, skip-layer connections and $\lambda = 0.01$) was selected by cross-validation on a single imputation of the training set, and we use a predictive approximation.[1] Table 2 shows the performance of this classifier on a test set of 301 complete examples and loss $L(j, k) = \mid j - k \mid$ for different strategies for dealing with the missing values. For imputation by Gibbs sampling we modelled each of the full conditionals using NOLR because all variables in this dataset are ordinal. Categorical inputs to models are put in as level indicators, so change corresponds to two indicators taking values $(0,0)$, $(1,0)$ and $(1,1)$. Throughout this example we predict age, emv and motor as categorical variables but treat them as continuous inputs to models. Models were selected by cross-validation based on the complete training examples only and used the predictive approximation described above. Several full conditionals benefited from a non-linear model.

We now classify 199 incomplete test examples using the classifier found in the last line of Table 2. Median imputation of missing values in the test set incurs loss 132 whereas unconditional imputation incurs loss 106. The Gibbs sampling imputation model incurs loss 91 and is predicting probabilities accurately (Figure 2). Michie *et al.* (1994) and references therein give alternative analyses of the head injury data.

NOLR has provided an interpretable network model for ordered classes, the missing data strategy successfully learns from incomplete training examples and classifies incomplete future examples, and the predictive approach is beneficial.

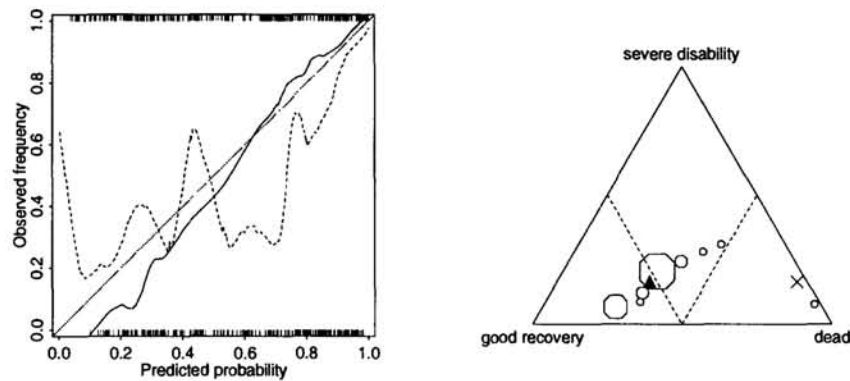

**Figure 2**: (a) Test set calibration for median imputation (dashed) and conditional imputation (solid). For predictions by conditional imputation we average $p(c \mid x^o, X^u)$ over 100 pseudo-independent samples from $p(x^u \mid x^o)$. Ticks on the lower (upper) axis denote predicted probabilities for the test examples using median (conditional) imputation. (b) In 100 pseudo-independent conditional imputations of the missing parts $x^u$ of a particular incomplete test example eight distinct values $x_i^u$ ($i = 1, \ldots, 8$) occur. (Recall that all components of $x$ are discrete.) For each distinct imputation we plot a circle with centre corresponding to $(p(1 \mid x^o, x_i^u), p(2 \mid x^o, x_i^u), p(3 \mid x^o, x_i^u))$ and area proportional to the number of occurrences of $x_i^u$ in the 100 imputations. The prediction by median imputation is located by $\times$; the average prediction over conditional imputations is located by $\blacktriangle$. Actual outcome is 'good recovery'. The conditional method correctly classifies the example and shows that the example is close to the Bayes decision boundary under loss $L(j, k) = \mid j - k \mid$ (dashed). Median imputation results in a confident and incorrect classification.

**Software:** A software library for fitting NOLR models in S-Plus is available at URL http://www.stats.ox.ac.uk/~mathies

**Acknowledgements:** The author thanks Brian Ripley for productive discussions of this work and Gordon Murray for permission to use the head injury dataset. This research was funded by the UK EPSRC and investment managers GMO Woolley Ltd.

## Footnotes

[1]For each completion $\mathcal{T}, \mathcal{X}_j^u$ of the training set we form a mixture approximation (7) to $p(\theta \mid \mathcal{T}, \mathcal{X}_j^u)$, sample from this 10,000 times and average the predicted probabilities. These predictions are averaged over completions. Maxima were found by running the optimizer 50 times from randomized starting weights. Up to 26 distinct maxima were found and approximately 5 generally accounted for over 95% of the posterior probability in most cases. Table 3 gives an example: averaging over maxima has greater effect than sampling around them, although both are useful. The cut-points $\phi$ in the NOLR model must satisfy order constraints, so we rejected samples of $\theta$ where these did not hold. However, the parameters were sufficiently well determined that this occurred in less than 0.5% of samples.

### References

Jennett, B., Teasdale, G., Braakman, R., Minderhoud, J., Heiden, J. & Kurze, T. (1979) Prognosis of patients with severe head injury. *Neurosurgery,* **4** 782–790.

Li, K.-H. (1988) Imputation using Markov chains. *Journal of Statistical Computation and Simulation,* **30** 57–79.

Little, R. & Rubin, D. B. (1987) Statistical Analysis with Missing Data. (Wiley, New York).

Mathieson, M. J. (1996) Ordinal models for neural networks. In *Neural Networks in Financial Engineering,* eds A.-P. Refenes, Y. Abu-Mostafa, J. Moody and A. S. Weigend (World Scientific, Singapore) 523–536.

McCullagh, P. (1980) Regression models for ordinal data. *Journal of the Royal Statistical Society Series B,* **42** 109–142.

Michie, D., Spiegelhalter, D. J. & Taylor, C. C. (eds) (1994) Machine Learning, Neural and Statistical Classification. (Ellis Horwood, New York).

Ripley, B. D. (1994) Flexible non-linear approaches to classification. In *From Statistics to Neural Networks. Theory and Pattern Recognition Applications,* eds V. Cherkassky, J. H. Friedman and H. Wechsler (Springer Verlag, New York) 108–126.

Ripley, B. D. (1996) *Pattern Recognition and Neural Networks.* (Cambridge University Press, Cambridge).